# Asymptotics of Gradient-based Neural Network Training Algorithms

**Sayandev Mukherjee**
saymukh@ee.cornell.edu
School of Electrical Engineering
Cornell University
Ithaca, NY 14853

**Terrence L. Fine**
tlfine@ee.cornell.edu
School of Electrical Engineering
Cornell University
Ithaca, NY 14853

## Abstract

We study the asymptotic properties of the sequence of iterates of weight-vector estimates obtained by training a multilayer feedforward neural network with a basic gradient-descent method using a fixed learning constant and no batch-processing. In the one-dimensional case, an exact analysis establishes the existence of a limiting distribution that is not Gaussian in general. For the general case and small learning constant, a linearization approximation permits the application of results from the theory of random matrices to again establish the existence of a limiting distribution. We study the first few moments of this distribution to compare and contrast the results of our analysis with those of techniques of stochastic approximation.

## 1 INTRODUCTION

The wide applicability of neural networks to problems in pattern classification and signal processing has been due to the development of efficient gradient-descent algorithms for the supervised training of multilayer feedforward neural networks with differentiable node functions. A basic version uses a fixed learning constant and updates all weights after each training input is presented (on-line mode) rather than after the entire training set has been presented (batch mode). The properties of this algorithm as exhibited by the sequence of iterates are not yet well-understood. There are at present two major approaches.

Stochastic approximation techniques (Bucklew,Kurtz,Sethares, 1993; Finnoff, 1993; Kuan,Hornik, 1991; White, 1989) study the limiting behavior of the stochastic process that is the piecewise-constant or piecewise-linear interpolation of the sequence of weight-vector iterates (assuming infinitely many i.i.d. training inputs) as the learning constant approaches zero. It can be shown (Bucklew,Kurtz,Sethares, 1993; Finnoff, 1993) that as the learning constant tends to zero, the fluctuation between the paths and their limit, suitably normalized, tends to a Gaussian diffusion process.

Leen and Moody (1993) and Orr and Leen (1993) have considered the Markov process formed by the sequence of iterates (again, assuming infinitely many i.i.d. training inputs) for a fixed nonzero learning constant. This approach has the merit of dealing with the nonzero learning constant case and of linking the study of the training algorithm with the well-developed literature on Markov processes.

In particular, it is possible to solve (Leen,Moody, 1993) for the asymptotic distribution of the sequence of weight-vector iterates from the Chapman-Kolmogorov equation after certain assumptions have been used to simplify it considerably. However, the assumptions are unrealistic: in particular, the assumption of detailed balance does not hold in more than one dimension. This approach also fails to establish the existence of a limiting distribution in the general case.

This paper follows the method of considering the sequence of weight-vector iterates as a discrete-time continuous state-space Markov process, when the learning constant is fixed and nonzero. We shall first seek to establish the existence of an asymptotic distribution, and then examine this distribution through its first few moments.

It can be proved (Mukherjee, 1994), using Foster's criteria (Tweedie, 1976) for the positive-recurrence of a Markov process, that when a single sigmoidal node with one parameter is trained using the iterative form of the basic gradient-descent training algorithm (without batch-processing), the sequence of iterates of the parameter has a limiting distribution which is in general non-Gaussian, thereby qualifying the oft-stated claims in the literature (see, for example, (Bucklew,Kurtz,Sethares, 1993; Finnoff, 1993; White, 1989)). However, this method proves to be intractable in the multiple parameter case.

## 2   THE GENERAL CASE AND LINEARIZATION IN $\underline{W}_n$

The general version of this problem for a neural network $\eta$ with scalar output involves training $\eta$ with the i.i.d. training sequence $\{(\underline{X}_n, Y_n)\}$, loss function $\mathcal{E}(\underline{x}, y, \underline{w}) = \frac{1}{2}[y - \eta(\underline{x}, \underline{w})]^2$ ($\underline{x} \in \mathbb{R}^d, y \in \mathbb{R}, \underline{w} \in \mathbb{R}^m$) and the gradient-descent updating equation for the estimates of the weight vector given by

$$\begin{aligned}\underline{W}_{n+1} &= \underline{W}_n - \mu \nabla_{\underline{w}} \mathcal{E}(\underline{x}, y, \underline{w})|_{(\underline{X}_{n+1}, Y_{n+1}, \underline{W}_n)} \\ &= \underline{W}_n + \mu[Y_{n+1} - \eta(\underline{W}_n, \underline{X}_{n+1})]\nabla_{\underline{w}}\eta(\underline{w}, \underline{x})|_{\underline{W}_n, \underline{X}_{n+1}}.\end{aligned}$$

As is customary in this kind of analysis, the training set is assumed infinite, so that $\{\underline{W}_n\}_{n=0}^{\infty}$ forms a homogeneous Markov process in discrete time. In our analysis, the training data is assumed to come from the model

$$Y = \eta(\underline{w}^0, \underline{X}) + Z,$$

where $Z$ and $\underline{X}$ are independent, and $Z$ has zero mean and variance $\sigma^2$. Hence, the unrestricted Bayes estimator of $Y$ given $\underline{X}$, $\mathbb{E}(Y|\underline{X}) = \eta(\underline{w}^0, \underline{X})$, is in the class of neural network estimators, and $\underline{w}^0$ is the goal of training. For convenience, we define $\underline{\tilde{W}} = \underline{W} - \underline{w}^0$.

Assuming that $\mu$ is small and that after a while, successive iterates, with high probability, jitter about in a close neighborhood of the optimal value $\underline{w}^0$, we make the important assumption that

$$\underline{\tilde{W}}_n = O_{\mathbf{P}}(\mu^k) \tag{1}$$

for some $0 < k < 1$ (see Section 4) [1]. Applying Taylor series expansions to $\eta$ and $\nabla_{\underline{w}}\eta$ and neglecting all terms $O_{\mathbf{P}}(\mu^{1+2k})$ and higher, we obtain the following linearized form of the updating equation:

$$\underline{\tilde{W}}_{n+1} = \mathbf{A}_{n+1}\underline{\tilde{W}}_n + \underline{B}_{n+1}, \tag{2}$$

where

$$\underline{B}_{n+1} = \mu Z_{n+1}\nabla_{\underline{w}}\eta(\underline{w}, \underline{x})|_{\underline{w}^0, \underline{X}_{n+1}},$$

$$
\begin{aligned}
\mathbf{A}_{n+1} &= \mathsf{I}_m - \mu(\nabla_{\underline{w}}\eta(\underline{w}, \underline{x})|_{\underline{w}^0, \underline{X}_{n+1}})(\nabla_{\underline{w}}\eta(\underline{w}, \underline{x})|_{\underline{w}^0, \underline{X}_{n+1}})^T \\
&\quad + \mu Z_{n+1}\nabla_{\underline{w}}\nabla_{\underline{w}}\eta(\underline{w}, \underline{x})|_{\underline{w}^0, \underline{X}_{n+1}} \tag{3} \\
&= \mathsf{I}_m - \mu(\mathbf{G}_{n+1} - Z_{n+1}\mathbf{J}_{n+1}), \tag{4}
\end{aligned}
$$

$$\mathbf{G}_{n+1} = (\nabla_{\underline{w}}\eta(\underline{w}, \underline{x})|_{\underline{w}^0, \underline{X}_{n+1}})(\nabla_{\underline{w}}\eta(\underline{w}, \underline{x})|_{\underline{w}^0, \underline{X}_{n+1}})^T$$

$$\mathbf{J}_{n+1} = (\nabla_{\underline{w}}\nabla_{\underline{w}}\eta(\underline{w}, \underline{x})|_{\underline{w}^0, \underline{X}_{n+1}})$$

do not depend on $\underline{W}_n$. The matrices $\{(\mathbf{A}_{n+1}, \underline{B}_{n+1})\}$ form an i.i.d. sequence, but $\mathbf{A}_{n+1}$ and $\underline{B}_{n+1}$ are dependent for each $n$. Hence the linearized $\underline{W}_n$ again forms a homogeneous Markov process in discrete time.

In what follows we analyze this process in the hope that its asymptotics agree with those of the original Markov process.

## 3   EXISTENCE OF A LIMITING DISTRIBUTION

Let $\mathbf{A}$, $\underline{B}$, $\mathbf{G}$, $\mathbf{J}$ denote random matrices with the common distributions of the i.i.d. sequences $\{\mathbf{A}_n\}$, $\{\underline{B}_n\}$, $\{\mathbf{G}_n\}$, and $\{\mathbf{J}_n\}$ respectively, and let $\mathbf{T} : \mathbb{R}^m \to \mathbb{R}^m$ be the random affine transformation

$$\underline{\tilde{w}} \mapsto \mathbf{A}\underline{\tilde{w}} + \underline{B}.$$

The following result establishes the existence of a limiting distribution of $\underline{W}_n$.

**Lemma 1 (Berger Thm. V, p.162)** *Suppose*

$$\mathbb{E}[\log^+ \|\mathbf{A}\| + \log^+ \|\underline{B}\|] \quad < \quad \infty; \tag{5}$$

$$\mathbb{E}\log\|\mathbf{A}_n\mathbf{A}_{n-1}\cdots\mathbf{A}_1\| \quad < \quad 0 \text{ *for some* } n \tag{6}$$

*where*

$$\log^+ x = \log x \vee 0.$$

*Then the following conclusions hold:*

1. *Unique stationary distribution: There exists a unique random variable $\underaccent{\tilde}{W} \in \mathbb{R}^m$, upto distribution, that is stationary with respect to $\mathbf{T}$ (i.e., $\underaccent{\tilde}{W}$ is independent of $\mathbf{T}$, and $\mathbf{T}\underaccent{\tilde}{W}$ has the same distribution as $\underaccent{\tilde}{W}$).*

2. *Asymptotic stationarity: We have convergence in distribution:*

$$\underaccent{\tilde}{W}_n \xrightarrow{\mathcal{D}} \underaccent{\tilde}{W}.$$

Our choice of norm is the operator norm for the matrix $\mathbf{A}$,

$$\|\mathbf{A}\| = \max |\lambda(\mathbf{A})|,$$

where $\{\lambda(\mathbf{A})\}$ are the eigenvalues of $\mathbf{A}$, and the Euclidean norm for the vector $\underaccent{\tilde}{B}$,

$$\|\underaccent{\tilde}{B}\| = \sqrt{\sum_{i=1}^{m} |B_i|^2}.$$

We first verify (5). From the inequality $\forall x \in \mathbb{R}$, $\log^+ x \leq x^2$, it is easily seen that if $\eta$ is a feedforward net where all activation functions are twice-continuously differentiable in the weights, all hidden-layer activation functions are bounded and have bounded derivatives up to order 2, and if the training sequence $(\underaccent{\tilde}{X}_n, Y_n)$ is i.i.d. with finite fourth moments, then (5) holds for the Euclidean norm for $\underaccent{\tilde}{B}$ and the Frobenius norm for $\mathbf{A}$, $\|\mathbf{A}\|^2 = \sum_{i=1}^{m}\sum_{j=1}^{m} |A_{ij}|^2$. Since

$$(\max |\lambda(\mathbf{A})|)^2 \leq \sum |\lambda(\mathbf{A})|^2 \leq \sum_{i=1}^{m}\sum_{j=1}^{m} |A_{ij}|^2, \tag{7}$$

we see that (5) also holds for the operator norm of $\mathbf{A}$.

Assumption (6) forces the product $\mathbf{A}_n \cdots \mathbf{A}_1$ to tend to $\mathbf{0}^{m \times m}$ almost surely (Berger, 1993, p.146) and therefore removes the dependence of the asymptotic distribution of $\{\underaccent{\tilde}{W}_n\}$ on that of the initial value $\underaccent{\tilde}{W}_0$. A sufficient condition for (6) is given by the following lemma.

**Lemma 2** *Suppose $\mathbb{E}\mathbf{G}$ is positive definite (note that it is positive semidefinite by definition), and for all $n$, $\mathbb{E}\mathbf{A}^n < \infty$. Then (6) holds for sufficiently small, positive $\mu$.*

*Proof:* By assumption, $\min \lambda(\mathbb{E}\mathbf{G}) = \delta > 0$ for some $\delta$.

Let $\mathbf{H}_n = \frac{1}{n}\sum_{i=1}^{n}(\mathbf{G}_i - Z_i\mathbf{J}_i)$. By the Strong Law of Large Numbers applied to the i.i.d. random matrices $(\mathbf{G}_i - Z_i\mathbf{J}_i)$, we have $\mathbf{H}_n \to \mathbb{E}\mathbf{G}$ a.s., so

$$\min \lambda(\mathbf{H}_n) \to \min \lambda(\mathbb{E}\mathbf{G}) \text{ a.s.} \tag{8}$$

Applying (7) to $\min \lambda(\mathbf{H}_n)$, it is easily shown that the same conditions on $\eta$ and the training sequence that are sufficient for (5) also give $\sup_n \mathbb{E}[\min \lambda(\mathbf{H}_n)]^2 < \infty$, which in turn implies that $\{\min \lambda(\mathbf{H}_n)\}$ are uniformly integrable. Together with (8), this implies (Loéve, 1977, p.165) that $\min \lambda(\mathbf{H}_n) \to \min \lambda(\mathbb{E}\mathbf{G})$ in $L^1$. Hence there

exists some (nonrandom) $N$, say, such that $\mathbb{E}|\min \lambda(\mathbf{H}_N) - \min \lambda(\mathbb{E}\mathbf{G})| \leq \delta/2$. Since

$$|\mathbb{E}\min \lambda(\mathbf{H}_N) - \min \lambda(\mathbb{E}\mathbf{G})| \leq \mathbb{E}|\min \lambda(\mathbf{H}_N) - \min \lambda(\mathbb{E}\mathbf{G})| \leq \delta/2,$$

we therefore have

$$\mathbb{E}\left[\min \lambda\left(\frac{1}{N}\sum_{i=1}^{N}(\mathbf{G}_i - Z_i\mathbf{J}_i)\right)\right] \geq \min \lambda(\mathbb{E}\mathbf{G}) - \delta/2 = \delta - \delta/2 = \delta/2 > 0. \quad (9)$$

We shall prove that (6) holds for this $N$ ($\geq m$) by showing that

$$\mathbb{E}\log \|\mathbf{A}_N\mathbf{A}_{N-1}\cdots\mathbf{A}_1\|^2 = 2\mathbb{E}\log \|\mathbf{A}_N\mathbf{A}_{N-1}\cdots\mathbf{A}_1\| < 0.$$

For our choice of norm, we therefore want $\mathbb{E}[(\log(\max|\lambda(\mathbf{A}_N\cdots\mathbf{A}_1)|))^2] < 0$. From Jensen's inequality, it is sufficient to have $\log \mathbb{E}[\max|\lambda(\mathbf{A}_N\cdots\mathbf{A}_1)|]^2 < 0$, or equivalently,

$$\mathbb{E}[\max|\lambda(\mathbf{A}_N\cdots\mathbf{A}_1)|]^2 < 1. \quad (10)$$

Now, since $N$ is fixed, we can choose $\mu$ small enough that

$$\mathbf{A}_N\cdots\mathbf{A}_1 = \mathsf{I}_m - \mu\sum_{i=1}^{N}(\mathbf{G}_i - Z_i\mathbf{J}_i) + O_\mathbf{P}(\mu^2).$$

Hence, $\lambda(\mathbf{A}_N\cdots\mathbf{A}_1) = 1 - \mu\lambda(\sum_{i=1}^{N}(\mathbf{G}_i - Z_i\mathbf{J}_i)) + O_\mathbf{P}(\mu^2)$, and $N$ is fixed, so

$$|\lambda(\mathbf{A}_N\cdots\mathbf{A}_1)|^2 = 1 - 2\mu\lambda\left(\sum_{i=1}^{N}(\mathbf{G}_i - Z_i\mathbf{J}_i)\right) + O_\mathbf{P}(\mu^2),$$

giving

$$\max|\lambda(\mathbf{A}_N\cdots\mathbf{A}_1)|^2 \leq 1 - 2\mu\min \lambda\left(\sum_{i=1}^{N}(\mathbf{G}_i - Z_i\mathbf{J}_i)\right) + O_\mathbf{P}(\mu^2),$$

$$\mathbb{E}\max|\lambda(\mathbf{A}_N\cdots\mathbf{A}_1)|^2 \leq 1 - N\delta\mu + o(\mu), \quad (11)$$

where we use (9) and the observation that the structure of the last $O_\mathbf{P}(\mu^2)$ term is such that its expectation (guaranteed finite by the hypothesis $\mathbb{E}A^N < \infty$) is $O(\mu^2)$, or $o(\mu)$, and we also restrict $\mu < 1/N\delta$ so that

$$1 - 2\mu\mathbb{E}\min \lambda\left(\sum_{i=1}^{N}(\mathbf{G}_i - Z_i\mathbf{J}_i)\right) > 0.$$

From (11), it is clear that (10) holds for all sufficiently small, positive $\mu$ ($\ll 1/N\delta$). Therefore (6) holds for $n = N$.

We can combine these two lemmas into the following theorem.

**Theorem 1** *Let $\eta$ be a feedforward net where all activation functions are twice-continuously differentiable in the weights, all hidden-layer activation functions are bounded and have bounded derivatives up to order 2, and let the training sequence $(\underline{X}_n, Y_n)$ be i.i.d. with finite moments. Further, assume that $\mathbb{E}\mathbf{G}$ is positive definite. Then, for all sufficiently small, positive $\mu$ the sequence of random vectors $\{\tilde{\underline{W}}_n\}_{n=1}^{\infty}$ obtained from the updating equation (2) has a unique limiting distribution.*

We circumvent the generally intractable problem of finding the limiting distribution by calculating and investigating the behavior of its moments.

## 4  MOMENTS OF THE LIMITING DISTRIBUTION

Let us assume that the mean and variance of the limiting distribution exist, and that $Z \sim \mathcal{N}(0, \sigma^2)$. From (2) and the form of $\mathbf{A}_{n+1}$ and $\underline{B}_{n+1}$, it is easy to show that $\mathbb{E}\tilde{W} = 0$, or $\mathbb{E}\underline{W} = \underline{w}^0$, so the optimal value $\underline{w}^0$ is the mean of the limiting distribution of the sequence of iterates $\{\underline{W}_n\}$. It can also be shown (Mukherjee, 1994) that $\mathbb{E}\underline{\tilde{W}}\underline{\tilde{W}}^T = (\mu\sigma^2/2)\mathsf{I}_m$, yielding $\underline{\tilde{W}} = O_{\mathbf{P}}(\sqrt{\mu})$. This is consistent with our assumption (1) with $k = 1/2$.

In the one-dimensional case ($d = m = 1$), we have $\mathbb{E}\tilde{W} = 0$ and $\mathbb{E}\tilde{W}^2 = \frac{1}{2}\mu\sigma^2$ if $\mathbb{E}[X_{n+1}\eta'(w^0 X_{n+1})]^2 \neq 0$. Using these results, the fact that $Z \sim \mathcal{N}(0, \sigma^2)$, $\mathbb{E}\tilde{W} = 0$, the independence of $Z$ and $X$, and assuming that $\mathbb{E}X^8 < \infty$, it is not difficult to compute the expressions

$$\mathbb{E}\tilde{W}^3 = \frac{\mu^2\sigma^4\mathbb{E}[X^3\eta''\eta'(1 - \mu X^2\eta'^2)]}{\mathbb{E}[X^2\eta'^2 - \mu X^4(\eta'^4 + \sigma^2\eta''^2) + \mu^2 X^6\eta'^2(\eta'^4/3 + \sigma^2\eta''^2)]},$$

and

$$\mathbb{E}\tilde{W}^4 = 3(\mathbb{E}\tilde{W}^2)^2 K_1(\mu) + \mathbb{E}\tilde{W}^3 K_2(\mu),$$

where

$$K_1(\mu) = \frac{\mathbb{E}[X^2\eta'^2(1 - \mu X^2\eta'^2)^2 + \mu X^4\eta'^4 + 3\mu^2 X^6\eta''^2\eta'^2]}{K(\mu)},$$

$$K_2(\mu) = \frac{18\mu^2\sigma^2\mathbb{E}[X^3\eta''\eta'(1 - \mu X^2\eta'^2)^2 + \mu^4\sigma^4 X^7\eta''^3\eta']}{K(\mu)},$$

$$K(\mu) = \mathbb{E}[X^2\eta'^2 - \frac{3}{2}\mu X^4(\eta'^4 + \sigma^2\eta''^2(1 - \mu X^2\eta'^2)^2)$$
$$+ \mu^2 X^6\eta'^6 - \frac{1}{4}\mu^3 X^8(\eta'^8 + 3\sigma^4\eta''^4)],$$

and $\eta'$ and $\eta''$ are evaluated at the argument $w^0 X$ for $\eta$.

From the above expressions, it is seen that if $\eta(\cdot) = 1/[1 + e^{-(\cdot)}]$ and $X$ has a symmetric distribution (say $\mathcal{N}(0, 1)$), then $\mathbb{E}\tilde{W}^3 \neq 0$ and $\mathbb{E}\tilde{W}^4 \neq 3(\mathbb{E}\tilde{W}^2)^2$, implying that $\tilde{W}$ is non-Gaussian in general. This result is consistent with that obtained by direct application of Foster's criterion (Mukherjee, 1994).

## 5  RECONCILING LINEARIZATION AND STOCHASTIC APPROXIMATION METHODS

The results of stochastic approximation analysis give a Gaussian distribution for $\tilde{W}$ in the limit as $\mu \to 0$ (Bucklew,Kurtz,Sethares, 1993; Finnoff, 1993). However, our results establish that the Gaussian distribution result is not valid for small nonzero $\mu$ in general. To reconcile these results, recall $\underline{\tilde{W}} = O_{\mathbf{P}}(\sqrt{\mu})$. Hence, if we consider

only moments of the normalized quantity $\tilde{W}/\sqrt{\mu}$ (and neglect higher-order terms in $O_{\mathbf{P}}(\sqrt{\mu})$), we obtain $\mathbb{E}(\tilde{W}/\sqrt{\mu})^3 = 0$ and $\mathbb{E}(\tilde{W}/\sqrt{\mu})^4 = 3[\mathbb{E}(\tilde{W}/\sqrt{\mu})^2]^2$, which suggests that the normalized quantity $\tilde{W}/\sqrt{\mu}$ is Gaussian in the limit of vanishing $\mu$, a conclusion also reached from stochastic approximation analysis.

In support of this theoretical indication that the conclusions of our analysis (based on linearization for small $\mu$) might tally with those of stochastic approximation techniques for small values of $\mu$, simulations were done on the simple one-dimensional training case of the previous section for 8 cases: $\mu = 0.1, 0.2, 0.3, 0.5$, and $\sigma^2 = 0.1, 0.5$ for each value of $\mu$, with $w^0$ fixed at 3. For each of the 8 cases, either 5 or 10 runs were made, with lengths (for the given values of $\mu$) of 810000, 500000, 300000, and 200000 respectively. Each run gave a pair of sequences $\{\tilde{W}_n\}$ obtained by starting off at $\tilde{W}_0 = 0$ and training the network independently twice. Each resulting sequence $\{\tilde{W}_n\}$ was then downsampled at a large enough rate that the true autocorrelation of the downsampled sequence was less than 0.05, followed by deleting the first 10% of the samples of this downsampled sequence, so as to remove any dependence on initial conditions that might persist. (Autocorrelation at lag unity for this Markov Chain was so high that when $\mu = 0.1$, a decimation rate of 9000 was required.) This was done to ensure that the elements of the resulting downsampled sequences could be assumed independent for the various hypothesis tests that were to follow.

(a) For each run of each case, the empirical distribution functions of the two downsampled sequences thus generated were compared by means of the Kolmogorov-Smirnov test (Bickel,Doksum, 1977) at level 0.95, with the null hypothesis being that both sequences had the same actual cumulative distribution function (assumed continuous). This test was passed with ease on all trials, thereby showing that a limiting distribution existed and was attained by such a training algorithm.

(b) For each run of each case, a skewness test and a kurtosis test (Bickel,Doksum, 1977) for normality were done at level 0.95 to test for normality. The sequences generated failed both tests for the $(\mu, \sigma)$ pair (0.1,0.1) and passed them both for the pairs (0.1,0.5), (0.3,0.1), (0.5,0.1), and (0.5,0.5). For the pair (0.2,0.5), the skewness test was passed and the kurtosis test failed, and for the pairs (0.2,0.1) and (0.3,0.5), the skewness test was failed and the kurtosis test passed.

(c) All trials cleared the Kolmogorov tests (Bickel,Doksum, 1977) for normality at level 0.95, both when the normal distribution was taken to have the sample mean and variance (computed on the downsampled sequence), and when the normal distribution function had the asymptotic values of mean (zero) and variance ($\mu\sigma^2/2$).

Hence we may conclude:

1. The limiting distribution of $\{W_n\}$ exists.

2. For small values of $\mu$, the deviation from Gaussianness is so small that the Gaussian distribution may be taken as a good approximation to the limiting distribution.

In other words, though stochastic approximation analysis states that $\tilde{W}/\sqrt{\mu}$ is Gaussian only in the limit of vanishing $\mu$, our simulation shows that this is a good approximation for small values of $\mu$ as well.

## Acknowledgements

The research reported here was partially supported by NSF Grant SBR-9413001.

## Footnotes

[1]I.e., $(\forall \epsilon > 0)(\exists M_\epsilon)(\forall n)\mathbb{P}(\mu^{-k}\|\underline{\tilde{W}}_n\| \le M_\epsilon) \ge 1 - \epsilon$.

## References

Berger, Marc A. *An Introduction to Probability and Stochastic Processes.* Springer-Verlag, New York, 1993.

Bickel, Peter, and Doksum, Kjell. *Mathematical Statistics: Basic Ideas and Selected Topics.* Holden-Day, San Francisco, 1977.

Bucklew, J.A., Kurtz, T.G., and Sethares, W.A. "Weak Convergence and Local Stability Properties of Fixed Step Size Recursive Algorithms," *IEEE Trans. Inform. Theory,* vol. 39, pp. 966-978, 1993.

Finnoff, W. "Diffusion Approximations for the Constant Learning Rate Backpropagation Algorithm and Resistence to Local Minima." In Giles, C.L., Hanson, S.J., and Cowan, J.D., editors, *Advances in Neural Information Processing Systems 5.* Morgan Kaufmann Publishers, San Mateo CA, 1993, p.459 ff.

Kuan, C-M, and Hornik, K. "Convergence of Learning Algorithms with Constant Learning Rates," *IEEE Trans. Neural Networks,* vol. 2, pp. 484-488, 1991.

Leen, T.K., and Moody, J.E. "Weight Space Probability Densities in Stochastic Learning: I. Dynamics and Equilibria," *Adv. in NIPS 5,* Morgan Kaufmann Publishers, San Mateo CA, 1993, p.451 ff.

Loéve, M. *Probability Theory I*, 4th ed. Springer-Verlag, New York, 1977.

Mukherjee, Sayandev. *Asymptotics of Gradient-based Neural Network Training Algorithms.* M.S. thesis, Cornell University, Ithaca, NY, 1994.

Orr, G.B., and Leen, T.K. "Probability densities in stochastic learning: II. Transients and Basin Hopping Times," *Adv. in NIPS 5,* Morgan Kaufmann Publishers, San Mateo CA, 1993, p.507 ff.

Rumelhart, D.E., Hinton, G.E., and Williams, R.J. "Learning interval representations by error propagation." In D.E. Rumelhart and J.L. McClelland, editors, *Parallel Distributed Processing*, Ch. 8, MIT Press, Cambridge MA, 1985.

Tweedie, R.L. "Criteria for Classifying General Markov Chains," *Adv. Appl. Prob.,* vol. 8, 737-771, 1976.

White, H. "Some Asymptotic Results for Learning in Single Hidden-Layer Feedforward Network Models," *J. Am. Stat. Assn.,* vol. 84, 1003-1013, 1989.
